# Parameterising Feature Sensitive Cell Formation in Linsker Networks in the Auditory System

**Lance C. Walton**
University of Kent at Canterbury
Canterbury
Kent
England

**David L. Bisset**
University of Kent at Canterbury
Canterbury
Kent
England

## Abstract

This paper examines and extends the work of Linsker (1986) on self organising feature detectors. Linsker concentrates on the visual processing system, but infers that the weak assumptions made will allow the model to be used in the processing of other sensory information. This claim is examined here, with special attention paid to the auditory system, where there is much lower connectivity and therefore more statistical variability. On-line training is utilised, to obtain an idea of training times. These are then compared to the time available to pre-natal mammals for the formation of feature sensitive cells.

## 1 INTRODUCTION

Within the last thirty years, a great deal of research has been carried out in an attempt to understand the development of cells in the pathways between the sensory apparatus and the cortex in mammals. For example, theories for the development of feature detectors were forwarded by Nass and Cooper (1975), by Grossberg (1976) and more recently Obermayer et al (1990).

Hubel and Wiesel (1961) established the existence of several different types of feature sensitive cell in the visual cortex of cats. Various subsequent experiments have

shown that a considerable amount of development takes place before birth (i.e. without environmental input). This must either be dependent on a genetic predispostion for individual cells to develop in an appropriate way without external influence, or some low level rules sufficient to create the required cell morphologies in the presence of random action potentials.

Although there is a great deal of *a priori* information concerning axon growth and synapse arborisation (governed by chemical means in the brain), it is difficult to conceive of a biological system that could use genetic information to directly manipulate the spatial information about the pre-synaptic target with respect to the axon with which the synapse is made. However, there is considerable random activity in the sensory apparatus that could be used to effect synaptic development.

Various authors have constructed models that deal with different aspects of self-organisation of this kind and some have pointed out the value of these types of cells in pattern classification problems (Grossberg 1976), but either the biological plausibility of these models is questionable, or the subject of pre-natal development is not addressed (i.e. without environmental input).

In this paper, the networks of Linsker (1986) will be examined. Although these networks have been analysed quite extensively by Linsker, and also by Mackay and Miller (1990), the biological aspects of parameter ranges and choices have only been touched upon. It is our aim in this paper, to add further detail in this area by examining the one-dimensional case which represents the auditory pathways.

## 2   LINSKER NETWORKS

The network is based on a Multi Layer Perceptron, with feed forward connections in all layers, and lateral connections (inhibition and excitation) in higher layers. The neural outputs are sums of the weighted inputs, and the weights develop according to a constrained Hebbian Rule. Each layer is lettered for reference starting from A and subsequent layers are lettered B,C,D etc. The superscript M will be used to refer to an arbitrary layer, and L is used to refer to the previous layer. Each layer has a set of parameters which are the same for all neurons in that layer. Connectivity is random but is based on a Gaussian density distribution ($exp(-r^2/r_M^2)$), where $r_M$ is the arbor radius for layer M.

Each layer is a rectangular array of neurons (or vector of neurons for the one dimensional case). The layers are assumed to be large enough so that edge effects are not important or do not occur. Layers develop one at a time starting from the B layer. The A layer is an input layer, which is divided into boxes, within each of which activity is uniform. This is biologically realistic, since sensory neurons fan out to a number of cells (an average of 10 in the cochlea) each of which only take input from one sensory cell. Hence the input layer for the network acts like a layer of tonotopically organised neurons.

## 3   NETWORK DEVELOPMENT

The output of a neuron in layer M is given by

$$F_n^{M\pi} = R_a + R_b . \sum_j c_{nj} F_{pre(nj)}^{L\pi} \tag{1}$$

Where,

$\pi$ indexes a pattern presentation,

The subscript n is used to index the M layer neurons,

$R_a, R_b$ are layer parameters,

$F_{pre(nj)}^{L\pi}$ is the output of the L layer neuron which is pre-synaptic to the j'th input of the n'th M layer neuron.

The synaptic weights develop according to a constrained Hebbian learning rule,

$$(\triangle c_{ni})^\pi = k_a + k_b . (F_n^{M\pi} - F_0^M) . (F_{pre(ni)}^{L\pi} - F_0^L) \tag{2}$$

Where,

$(\triangle c_{ni})^\pi$ is the change in the i'th weight of neuron n,

$k_a, k_b, F_0^M, F_0^L$ are layer parameters.

Synaptic weights are constrained to lie within the range $(n_{em} - 1, n_{em})$. (In this work, $n_{em} = 0.5$)

Linsker (1986a) derives an Ensemble Averaged Development equation which shows how development depends on the parameters, and how correlations develop between spatially proximate neurons in layers beyond the first. In so doing, the number of parameters is reduced from five per layer to two per layer, and therefore the equation is a very useful aid in understanding the self-organising nature of this model. The development equation is

$$\dot{c}_{ni} = K_1 + K_2 . \bar{c}_n + \frac{\sum_j Q_{pre(ni).pre(nj)}^L . c_{nj}}{N_M} \tag{3}$$

$$Q_{ij}^L \equiv \frac{< (F_i^{L\pi} - \bar{F}^L).(F_j^{L\pi} - \bar{F}^L)}{f_0^2} \tag{4}$$

Where,

$N_M$ is the number of synaptic connections to an M layer neuron,

$\bar{F}^L$ is the average output activity in the L layer,

$$K_1 = \frac{k_a + k_b.(R_a - F_0^M).(\bar{F}^L - F_0^L)}{N_M k_b R_b f_0^2} \tag{5}$$

$$K_2 = \frac{\bar{F}^L.(\bar{F}^L - F_0^L)}{f_0^2} \tag{6}$$

$f_0^2$ is a unit of activity used to normalise the two point correlation function $Q_{ij}^L$. In this work $f_0^2$ is chosen to set $Q_{ii}^L = 1$

Angle brackets denote an average taken over the ensemble of input patterns.

## 4   MORPHOLOGICAL REGIMES

From equation 3, an expression can be found for the average weight value $\bar{c}$ in a layer, and therefore certain properties of the system can be described. Although Mackay and Miller (1990) have described the regimes with the aid of eigenvalues and eigenfunctions, there is a much simpler method which will provide the same information.

For an all-excitatory (AE) layer, the average weight value is equal to $n_{em}$. Since all weights are equal to $n_{em}$, the summation in equation 3 can be re-written $n_{em} \cdot \sum_j Q^L_{pre(ni).pre(nj)} = n_{em} \cdot N_M \cdot \bar{q}$, where $\bar{q} = \frac{r_B}{2 \cdot N_C \cdot \sqrt{r_C^2 + r_B^2}}$.

A similar expression can be found for all-inhibitory (AI) layers, and therefore the $K_1 - K_2$ plane can be sub-divided into three regions which will yield AE cells, AI cells, and mixed-mode cells (see figure 1).

The plane can be divided further for the mixed-mode cell type in the C layer. On-center and off-center cells develop close to the AE and AI boundaries respectively. Mackay and Miller have shown why these cells develop and have placed a theoretical lower bound on $\bar{c}$ which agrees with experimental data. However, in so doing the effect of the intercept on the $K_2$ axis was deemed small, due to a large number of synaptic connections. This approximation depends upon the large number of connections between the B and C layers. In the auditory case, the number of connections is smaller, and it is possible that this assumption no longer holds.

From equation 3, it can be seen that movement into the On-Centre region from the AE region, causes the value of $\sum_j Q^L_{pre(ni).pre(nj)} \cdot c_{nj}$ to decrease. This has the effect of moving the intercept of the constant $\bar{c}$ line from $K_2 = \bar{q}$ towards $K_2 = 0$. $K_2$ finally reaches 0 when $\bar{c} = 0$, and then begins to move back towards $\bar{q}$ as the AI regime is approached.

This has two potentially important effects. Firstly, it means that the tolerance of $K_2$ varies with $K_1$; for a particular value of $K_1$, there are upper and lower limits on the value of $K_2$ which will allow maturation of on-center cells. This range of values (i.e. the difference between the limits) varies in a linear way with $K_1$, but the ratio of the range to a value of $K_2$ which is within the range (i.e. the center value) is not linear with $K_1$. Here, tolerance is defined as that ratio. Secondly, there is a region of negative $K_2$ where the nature of the cell morphology which will be produced is unknown.

It is therefore important that $K_2$ should be larger than this value in order to produce On-Center or Off-Center cells reliably. Mackay and Miller use $|K_2| \to \infty$ in their analysis. Unfortunately, this would require the fundamental network parameter $F_0^L \to \infty$ from equation 6, and therefore is an unsuitable choice. It is reasonable to assume that $F_0^L$ is of the same order as $\bar{F}^L$, and hence an order for $K_2$ can be established. For a concrete example, assume inputs are binary (giving $Q^L_{ii} = 0.25$) and $F_0^L = \bar{F}^L \times 1.2$, this will ensure $K_2 < 0$ (equation 6) while adhering to the assumption made above. Equation 6 now gives the order for $K_2 = 0.2$.

To find the value of $\bar{q}$, which will place a lower bound on $|K_2|$, a particular system should be chosen. The auditory system is chosen here.

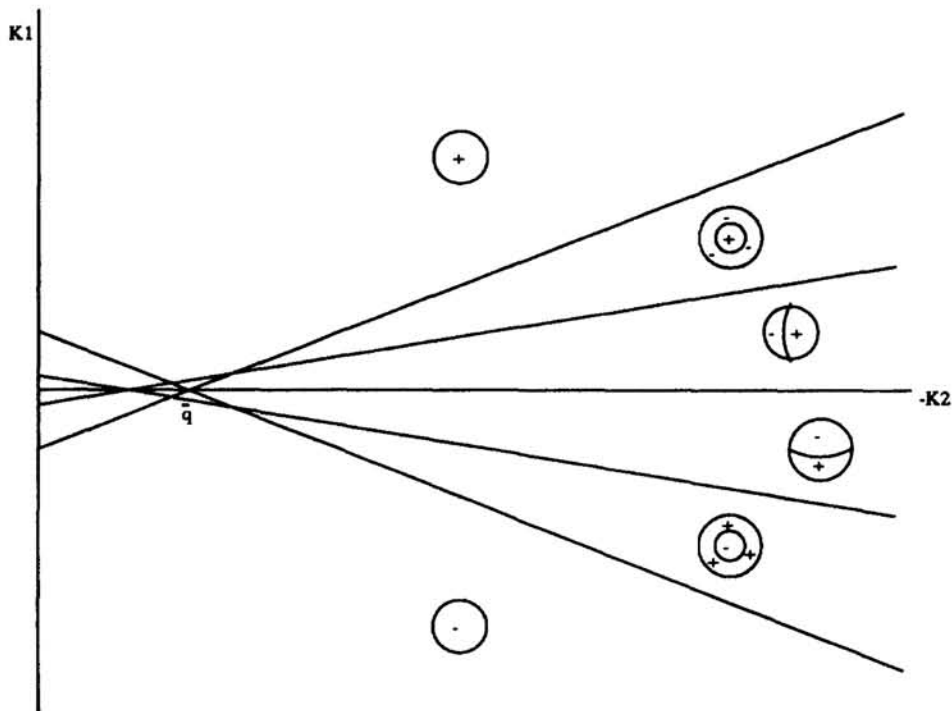

Figure 1: Graph of Morphological Regions for C Layer

There are approximately 3000 inner hair cells in the cochlea, each of which fans out to an average of 10 neurons (which sets our box size $\rho = 10$). These neurons take input from only one hair cell. The anteroventral cochlea nucleus takes input from this layer of cells, with a fan in $N_B \approx 50$ (c.f. the value of $N_B = 1000$ in Linsker (1986a)). The assumption is made that the three sections of the cochlea nucleus each contain approximately the same number of cells. With this smaller number of connections, the correlation function for this layer is somewhat coarser, and does not follow the theoretical curve for the continuum limit so well.

In addition, the on-center cells found in the posteroventral cochlea nucleus and the dorsal nucleus have centres with a tuning curve response Q of about 2.5 which corresponds to about 2000 B layer cells. If it is assumed that the surround of the cell is half the width of the core, then there is a total $r_C \approx 3000$ neurons. Simulations here use $N_C = 100$ which is a realistic number of connections in the context of a one-dimensional network.

In general, the arbor radius increases as layers become closer to the cortex. From Linsker, $r_C/r_B = 3$. $r_B$ is therefore equal to 1000. This yields the average number of connections to a given B cell from a particular A box being approximately unity, which agrees well with the condition expressed by Linsker.

Using the expression above, $\bar{q}$ can be calculated as approximately $1.5 \times 10^{-3}$. This value is certainly insignificant with respect to the value of $K_2 = 0.2$ quoted earlier, and therefore any effects due to the summation term in equation 3 can be ignored in the calculation of $\bar{c}$ for this system. This means that the original approximation still holds even in this low connectivity case.

## 5    SIMULATION RESULTS

A network was trained using the connectivity stated above to give various values of $\bar{c}$ with $K_2 = 0.2$. To obtain an idea of the total number of presentations that were required to train the network, without any artifacts that might be produced as a result of batch training, the original network equations were used. In all of these simulations, $R_a, F_0^M = 0$ so that the value of $K_1$ could be easily controlled.

The findings were that the maximum value of $k_b$ was about $10^{-3}$ which required 2.5 million pattern presentations to mature the network. With this value, on-center cells with an average weight value less than about 0.3 would not mature. However as the value of $k_b$ was decreased (keeping $K_1$ constant), the value of $\bar{c}$ could be made lower, at the expense of more pattern presentations. The figures obtained for the maturation of feature sensitive cells are extremely biologically realistic in the light of the number of pattern presentations available to an average mammal. For example, the foetal cat has sufficient time for about 25 million presentations (assuming 10 presentations per second).

## 6    CONCLUSION

We have shown that the class of network developed by Linsker is extendable to the auditory system where the number and density of synapses is considerably smaller than in the visual case. It has also been shown that the time for layer maturation by this method is sufficiently short even for mammals with a relatively short gestation period, and therefore should also be sufficient in mammals with longer foetal development times. We conclude that the model is therefore a good representation of feature detector development in the pre-natal mammal.

### References

Grossberg  S. (1976) - On the Development of Feature Detectors in the Visual Cortex with Applications to Learning and Reaction Diffusion Systems, *Biological Cybernetics 21*, 145 - 159

Grossberg  S. (1976) - Adaptive Pattern Classification and Universal Recoding : 1 Parallel Development and Coding of Neural Feature Detectors, *Biological Cybernetics 23*, 121 - 134

Hubel  D. H. and Wiesel  T. N. (1961) - Receptive Fields, Binocular Interaction and Functional Architechture in the Cat's Visual Cortex, *Journal of Physiology, 160*, 106 - 154

Kalil  R. E. (1989) - Synapse Formation In The Developing Brain, *Scientific American, December 1989*, 38 - 45

Klinke  R. (1986) - Physiology of Hearing, In Schmidt  R. W. (ed.), *Fundamentals of Sensory Physiology*, 199 - 223

MacKay  D. J. C. and Miller  K. D. (1990) - Analysis of Linsker's Simulations of Hebbian Rules, *Neural Computation, 2*, 173 - 187

von der Malsburg  C. (1979) - Development of Ocularity Domains and Growth

Behaviour of Axon Terminals, *Biological Cybernetics, 32*, 49 - 62

Linsker R. (1986a) - From Basic Network Principles To Neural Architecture : Emergence Of Spatial-Opponent Cells, *Proceedings of the National Academy of Sciences (USA), 83*, 7508 - 7512

Linsker R. (1986b) - From Basic Network Principles To Neural Architecture : Emergence of Orientation-Selective Cells, *Proceedings of the National Academy of Sciences (USA), 83*, 8390 - 8394

Linsker R. (1986c) - From Basic Network Principles To Neural Architecture : Emergence of Orientation-Columns, *Proceedings of the National Academy of Sciences (USA), 83*, 8779 - 8783

Nass M. M. and Cooper L. N. (1975) - A Theory for the Development of Feature Detecting Cells in the Visual Cortex, *Biological Cybernetics, 19*, 1 - 18

Obermayer K. Ritter H. and Schulten K. (1990) - Development and Spatial Structure of Cortical Feature Maps: A Model Study *NIPS, 3*, 11 - 17

Sloman A. (1989) - On Designing a Visual System (Towards a Gibsonian Computational Model of Vision) *Journal of Experimental and Theoretical Artificial Intelligence, 1*, 289 - 337

Tanaka S. (1990) - Interaction among Ocularity, Retinotopy and On-Center/Off Center Pathways During Development *NIPS, 3*, 18 - 25